# Recursive Segmentation and Recognition Templates for 2D Parsing

**Long (Leo) Zhu**
CSAIL MIT
leozhu@csail.mit.edu

**Yuanhao Chen**
USTC
yhchen4@ustc.edu.cn

**Yuan Lin**
Shanghai Jiaotong University
loirey@sjtu.edu.cn

**Chenxi Lin**
Microsoft Research Asia
chenxil@microsoft.com

**Alan Yuille**
UCLA
yuille@stat.ucla.edu

## Abstract

Language and image understanding are two major goals of artificial intelligence which can both be conceptually formulated in terms of parsing the input signal into a hierarchical representation. Natural language researchers have made great progress by exploiting the 1D structure of language to design efficient polynomial-time parsing algorithms. By contrast, the two-dimensional nature of images makes it much harder to design efficient image parsers and the form of the hierarchical representations is also unclear. Attempts to adapt representations and algorithms from natural language have only been partially successful.

In this paper, we propose a Hierarchical Image Model (HIM) for 2D image parsing which outputs image segmentation and object recognition. This HIM is represented by recursive segmentation and recognition templates in multiple layers and has advantages for representation, inference, and learning. Firstly, the HIM has a coarse-to-fine representation which is capable of capturing long-range dependency and exploiting different levels of contextual information. Secondly, the structure of the HIM allows us to design a rapid inference algorithm, based on dynamic programming, which enables us to parse the image rapidly in polynomial time. Thirdly, we can learn the HIM efficiently in a discriminative manner from a labeled dataset. We demonstrate that HIM outperforms other state-of-the-art methods by evaluation on the challenging public MSRC image dataset. Finally, we sketch how the HIM architecture can be extended to model more complex image phenomena.

## 1   Introduction

Language and image understanding are two major tasks in artificial intelligence. Natural language researchers have formalized this task in terms of parsing an input signal into a hierarchical representation. They have made great progress in both representation and inference (i.e. parsing). Firstly, they have developed probabilistic grammars (e.g. stochastic context free grammar (SCFG) [1] and beyond [2]) which are capable of representing complex syntactic and semantic language phenomena. For example, speech contains elementary constituents, such as nouns and verbs, that can be recursively composed into a hierarchy of (e.g. noun phrase or verb phrase) of increasing complexity. Secondly, they have exploited the one-dimensional structure of language to obtain efficient polynomial-time parsing algorithms (e.g. the inside-outside algorithm [3]).

By contrast, the nature of images makes it much harder to design efficient image parsers which are capable of simultaneously performing segmentation (parsing an image into regions) and recognition (labeling the regions). Firstly, it is unclear what hierarchical representations should be used to model images and there are no direct analogies to the syntactic categories and phrase structures that occur in speech. Secondly, the inference problem is formidable due to the well-known complexity

and ambiguity of segmentation and recognition. Unlike most languages (Chinese is an exception), whose constituents are well-separated words, the boundaries between different image regions are usually highly unclear. Exploring all the different image partitions results in combinatorial explosions because of the two-dimensional nature of images (which makes it impossible to order these partitions to enable dynamic programming). Overall it has been hard to adapt methods from natural language parsing and apply them to vision despite the high-level conceptual similarities (except for restricted problems such as text [4]).

Attempts at image parsing must make trade-offs between the complexity of the models and the complexity of the computation (for inference and learning). Broadly speaking, recent attempts can be divided into two different styles. The first style emphasizes the modeling problem and develops stochastic grammars [5, 6] capable of representing a rich class of visual relationships and conceptual knowledge about objects, scenes, and images. This style of research pays less attention to the complexity of computation. Learning is usually performed, if at all, only for individual components of the models. Parsing is performed by MCMC sampling and is only efficient provided effective proposal probabilities can be designed [6]. The second style builds on the success of conditional random fields (CRF's) [7] and emphasizes efficient computation. This yields simpler (discriminative) models which are less capable of representing complex image structures and long range interactions. Efficient inference (e.g. belief propagation and graph-cuts) and learning (e.g. AdaBoost, MLE) are available for basic CRF's and make these methods attractive. But these inference algorithms become less effective, and can fail, if we attempt to make the CRF models more powerful. For example, TextonBoost [8] requires the parameters of the CRF to be tuned manually. Overall, it seems hard to extend the CRF style methods to include long-range relationships and contextual knowledge without significantly altering the models and the algorithms.

In this paper, we introduce Hierarchical Image Models (HIM)'s for image parsing. HIM's balance the trade-off between model and inference complexity by introducing a hierarchy of hidden states. In particular, we introduce *recursive segmentation and recognition templates* which represent complex image knowledge and serve as elementary constituents analogous to those used in speech. As in speech, we can recursively compose these constituents at lower levels to form more complex constituents at higher level. Each node of the hierarchy corresponds to an image region (whose size depends on the level in the hierarchy). The state of each node represents both the partitioning of the corresponding region into segments and the labeling of these segments (i.e. in terms of objects). Segmentations at the top levels of the hierarchy give coarse descriptions of the image which are refined by the segmentations at the lower levels. Learning and inference (parsing) are made efficient by exploiting the hierarchical structure (and the absence of loops). In short, this novel architecture offers two advantages: (I) Representation – the hierarchical model using segmentation templates is able to capture long-range dependency and exploiting different levels of contextual information, (II) Computation – the hierarchical tree structure enables rapid inference (polynomial time) and learning by variants of dynamic programming (with pruning) and the use of machine learning (e.g. structured perceptrons [9]).

To illustrate the HIM we implement it for parsing images and we evaluate it on the public MSRC image dataset [8]. Our results show that the HIM outperforms the other state-of-the-art approaches. We discuss ways that HIM's can be extended naturally to model more complex image phenomena.

## 2 Hierarchical Image Model

### 2.1 The Model

We represent an image by a hierarchical graph defined by parent-child relationships. See figure 1. The hierarchy corresponds to the image pyramid (with 5 layers in this paper). The top node of the hierarchy represents the whole image. The intermediate nodes represent different sub-regions of the image. The leaf nodes represent local image patches ($27 \times 27$ in this paper). We use $a$ to index nodes of the hierarchy. A node $a$ has only one parent node denoted by $Pa(a)$ and four child nodes denoted by $Ch(a)$. Thus, the hierarchy is a quad tree and $Ch(a)$ encodes all its vertical edges. The image region represented by node $a$ is denoted by $R(a)$. A pixel in $R(a)$, indexed by $r$, corresponds to an image pixel. The set of pairs of neighbor pixels in $R(a)$ is denoted by $E(a)$.

A configuration of the hierarchy is an assignment of state variables $y = \{y_a\}$ with $y_a = (s_a, c_a)$ at each node $a$, where $s$ and $c$ denote region partition and object labeling, respectively and $(s, c)$ is called the "Segmentation and Recognition" pair, which we call an *S-R pair*. All state variables are

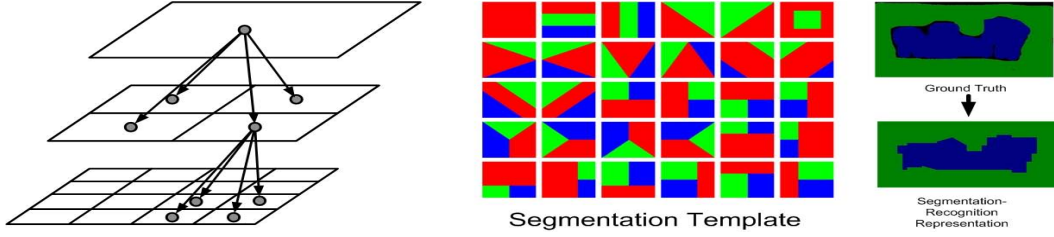

**Figure 1:** The left panel shows the structure of the Hierarchical Image Model. The grey circles are the nodes of the hierarchy. All nodes, except the top node, have one parent nodes. All nodes except the leafs are connected to four child nodes. The middle panel shows a dictionary of 30 segmentation templates. The color of the sub-parts of each template indicates the object class. Different sub-parts may share the same label. For example, three sub-parts may have only two distinct labels. The last panel shows that the ground truth pixel labels (upper right panel) can be well approximated by composing a set of labeled segmentation templates (bottom right panel).

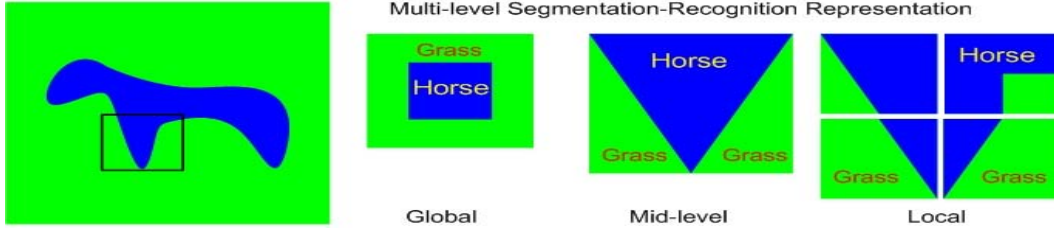

**Figure 2:** This figure illustrates how the segmentation templates and object labels (S-R pair) represent image regions in a coarse-to-fine way. The left figure is the input image which is followed by global, mid-level and local S-R pairs. The global S-R pair gives a coarse description of the object identity (horse), its background (grass), and its position in the image (central). The mid-level S-R pair corresponds to the region bounded by the black box in the input image. It represents (roughly) the shape of the horse's leg. The four S-R pairs at the lower level combine to represent the same leg more accurately.

unobservable. More precisely, each region $R(a)$ is described by a *segmentation templates* which is selected from a dictionary $D_S$. Each segmentation template consists of a partition of the region into $K$ non-overlapping sub-parts, see figure 1. In this paper $K \leq 3$, $|D_s| = 30$, and the segmentation templates are designed by hand to cover the taxonomy of shape segmentations that happen in images, such as T-junctions, Y-junctions, and so on. The variable $s$ refers to the indexes of the segmentation templates in the dictionary, i.e., $s_a \in \{1..|D_s|\}$. $c$ gives the object labels of $K$ sub-parts (i.e. labels one sub-part as "horse" another as "dog" and another as "grass"). Hence $c_a$ is a K-dimension vector whose components take values $1, ..., M$ where $M$ is the number of object classes. The labeling of a pixel $r$ in region $R(a)$ is denoted by $o_a^r \in \{1..M\}$ and is directly obtained from $s_a, c_a$. Any two pixels belonging to the same sub-part share the same label. The labeling $o_a^r$ is defined at the level of node $a$. In other words, each level of the hierarchy has a separate labeling field. We will show how our model encourages the labelings $o_a^r$ at different levels to be consistent.

A novel feature of this hierarchical representation is the multi-level *S-R pairs* which explicitly model both the segmentation and labeling of its corresponding region, while traditional vision approaches [8, 10, 11] use labeling only. The S-R pairs defined in a hierarchical form provide a coarse-to-fine representation which captures the "gist" (semantical meaning) of image regions. As one can see in figure 2, the global S-R pair gives a coarse description (the identities of objects and their spatial layout) of the whole image which is accurate enough to encode high level image properties in a compact form. The mid-level one represents the leg of a horse roughly. The four templates at the lower level further refine the interpretations. We will show this approximation quality empirically in section 3.

The conditional distribution over all the states is given by:

$$p(y|x;\alpha) = \frac{1}{Z(x;\alpha)} \exp\{-E_1(x,s,c;\alpha_1) - E_2(x,s,c;\alpha_2) - E_3(s,c;\alpha_3) \quad (1)$$
$$-E_4(c;\alpha_4) - E_5(s;\alpha_5) - E_6(s,c;\alpha_6)\}$$

where $x$ refers to the input image, $y$ is the parse tree, $\alpha$ are the parameters to be estimated, $Z(x;\alpha)$ is the partition function and $E_i(x,y)$ are energy terms. Equivalently, the conditional distribution can be reformulated in a log-linear form:

$$\log p(y|x;\alpha) = \psi(x,y) \cdot \alpha - \log Z(x;\alpha) \quad (2)$$

Each energy term is of linear form, $E_i(x, y) = -\psi_i(x, y) \cdot \alpha_i$, where the inner product is calculated on potential functions defined over the hierarchical structure. There are six types of energy terms defined as follows.

The first term $E_1(x, s, c)$ is an object specific data term which represents image features of regions. We set $E_1(x, s, c) = -\sum_a \alpha_1 \psi_1(x, s_a, c_a)$ where $\sum_a$ is the summation over all nodes at different levels of the hierarchy, and $\psi_1(x, s_a, c_a)$ is of the form:

$$\psi_1(x, s_a, c_a) = \frac{1}{|R(a)|} \sum_{r \in R(a)} \log p(o_a^r | x) \tag{3}$$

where $p(o_a^r | x) = \frac{\exp\{F(x^r, o_a^r)\}}{\sum_{o'} \exp\{F(x^r, o')\}}$, $x^r$ is a local image region centered at the location of $r$, and $F(\cdot, \cdot)$ is a strong classifier output by multi-class boosting [12]. The image features used by the classifier (47 in total) are the greyscale intensity, the color (R,G, B channels), the intensity gradient, the Canny edge, the response of DOG (difference of Gaussians) and DOOG (Difference of Offset Gaussian) filters at different scales (13*13 and 22*22) and orientations (0,30,60,...), and so on. We use 55 types of shape (spatial) filters (similar to [8]) to calculate the responses of 47 image features. There are $2585 = 47 * 55$ features in total.

The second term (segmentation specific) $E_2(x, s, c) = -\sum_a \alpha_2 \psi_2(x, s_a, c_a)$ is designed to favor the segmentation templates in which the pixels belonging to the same partitions (i.e., having the same labels) have similar appearance. We define:

$$\psi_2(x, s_a, c_a) = \frac{1}{|E(a)|} \sum_{(q,r) \in E(a)} \phi(x^r, x^q | o_a^r, o_a^q) \tag{4}$$

where $E(a)$ are the set of edges connecting pixels $q, r$ in a neighborhood and $\phi(x^r, x^q | o_a^r, o_a^q)$ has the form of $\phi(x^r, x^q | o_a^r, o_a^q) = \begin{cases} \gamma(r,q) & if \ o_a^r = o_a^q \\ 0 & if \ o_a^r \neq o_a^q \end{cases}$, where $\gamma(r, q) = \lambda \exp\{-\frac{g^2(r,q)}{2\gamma^2}\} \frac{1}{dist(r,q)}$, $g(.,.)$ is a distance measure on the colors $x^r, x^q$ and $dist(r, q)$ measures the spatial distance between $r$ and $q$. $\phi(x^r, x^q | o_a^r, o_a^q)$ is so called the contrast sensitive Potts model which is widely used in graph-cut algorithms [13] as edge potentials (only in one level) to favors pixels with similar colour having the same labels.

The third term, defined as $E_3(s, c) = -\sum_{a, b=Pa(a)} \alpha_3 \psi_3(s_a, c_a, s_b, c_b)$ (i.e. the nodes $a$ at all levels are considered and $b$ is the parent of $a$) is proposed to encourage the consistency between the configurations of every pair of parent-child nodes in two consecutive layers. $\psi_3(s_a, c_a, s_b, c_b)$ is defined by the Hamming distance:

$$\psi_3(s_a, c_a, s_b, c_b) = \frac{1}{|R(a)|} \sum_{r \in R(a)} \delta(o_a^r, o_b^r) \tag{5}$$

where $\delta(o_a^r, o_b^r)$ is the Kronecker delta, which equals one whenever $o_a^r = o_b^r$ and zero otherwise. The hamming function ensures to glue the segmentation templates (and their labels) at different levels together in a consistent hierarchical form. This energy term is a generalization of the interaction energy in the Potts model. However, $E_3(s, c)$ has a hierarchical form which allows multi-level interactions.

The fourth term $E_4(c)$ is designed to model the co-occurrence of two object classes (e.g., a cow is unlikely to appear next to an aeroplane):

$$E_4(c) = -\sum_a \sum_{i,j=1..M} \alpha_4(i,j) \psi_4(i, j, c_a, c_a) - \sum_{a, b=Pa(a)} \sum_{i,j=1..M} \alpha_4(i,j) \psi_4(i, j, c_a, c_b) \tag{6}$$

where $\psi_4(i, j, c_a, c_b)$ is an indicator function which equals one while $i \equiv c_a$ and $j \equiv c_b$ ($i \equiv c_a$ means $i$ is a component of $c_a$) hold true and zero otherwise. $\alpha_4$ is a matrix where each entry $\alpha_4(i, j)$ encodes the compatibility between two classes $i$ and $j$. The first term on the r.h.s encodes the classes in a single template while the second term encodes the classes in two templates of the parent-child nodes. It is worth noting that class dependency is encoded at all levels to capture both short-range and long-range interactions.

The fifth term $E_5(s) = -\sum_a \alpha_5 \psi_5(s_a)$, where $\psi_5(s_a) = \log p(s_a)$ encode the generic prior of the segmentation template. Similarly the sixth term $E_6(s,c) = -\sum_a \sum_{j \equiv c_a} \alpha_6 \psi_6(s_a, j)$, where $\psi_6(s_a, j) = \log p(s_a, j)$, models the co-occurrence of the segmentation templates and the object classes. $\psi_5(s_a)$ and $\psi_6(s_a, j)$ are directly obtained from training data by label counting. The parameters $\alpha_5$ and $\alpha_6$ are both scalars.

**Justifications.** The HIM has several partial similarities with other work. HIM is a coarse-to-fine representation which captures the "gist" of image regions by using the S-R pairs at multiple levels. But the traditional concept of "gist" [14] relies only on image features and does not include segmentation templates. Levin and Weiss [15] use a segmentation mask which is more object-specific than our segmentation templates (and they do not have a hierarchy). It is worth nothing that, in contrast to TextonBoost [8], we do not use "location features" in order to avoid the dangers of overfitting to a restricted set of scene layouts. Our approach has some similarities to some hierarchical models (which have two-layers only) [10],[11] – but these models also lack segmentation templates. The hierarchial model proposed by [16] is an interesting alternative but which does not perform explicit segmentation.

## 2.2 Parsing by Dynamic Programming

Parsing an image is performed as inference of the HIM. More precisely, the task of parsing is to obtain the maximum a posterior (MAP):

$$y^* = \arg\max_y p(y|x; \alpha) = \arg\max_y \psi(x, y) \cdot \alpha \tag{7}$$

The size of the states of each node is $O(M^K|D_s|)$ where $K = 3, M = 21, |D_s| = 30$ in our case. Since the form of $y$ is a tree, Dynamic Programming (DP) can be applied to calculate the best parse tree $y^*$ according to equation 7. Note that the pixel label $o_a$ is determined by $(s, c)$, so we only need consider a subset of pixel labelings. It is unlike flat MRF representation where we need to do exhaustive search over all pixel labels $o$ (which would be impractical for DP). The final output of the model for segmentation is the pixel labeling determined by the $(s, c)$ of the lowest level.

It is straight forward to see that the computational complexity of DP is $O(M^{2K}|D_s|^2H)$ where $H$ is the number of edges of the hierarchy. Although DP can be performed in polynomial time, the huge number of states make exact DP still impractical. Therefore, we resort to a pruned version of DP similar to the method described in [17]. For brevity we omit the details.

## 2.3 Learning the Model

Since HIM is a conditional model, in principle, estimation of its parameters can be achieved by any discriminative learning approach, such as maximum likelihood learning as used in Conditional Random Field (CRF) [7], max-margin learning [18], and structure-perceptron learning [9]. In this paper, we adopt the structure-perceptron learning which has been applied for learning the recursive deformable template (see paper [19]). Note that structure-perceptron learning is simple to implement and only needs to calculate the most probable configurations (parses) of the model. By contrast, maximum likelihood learning requires calculating the expectation of features which is difficult due to the large states of HIM. Therefore, structure-perceptron learning is more flexible and computationally simpler. Moreover, Collins [9] proved theoretical results for convergence properties, for both separable and non-separable cases, and for generalization.

The structure-perceptron learning will not compute the partition function $Z(x; \alpha)$. Therefore we do not have a formal probabilistic interpretation. The goal of structure-perceptron learning is to learn a mapping from inputs $x \in X$ to output structure $y \in Y$. In our case, $X$ is a set of images, with $Y$ being a set of possible parse trees which specify the labels of image regions in a hierarchical form. It seems that the ground truth of parsing trees needs all labels of both segmentation template and pixel labelings. In our experiment, we will show that how to obtain the ground truth directly from the segmentation labels without extra human labeling. We use a set of training examples $\{(x_i, y_i) : i = 1...n\}$ and a set of functions $\psi$ which map each $(x, y) \in X \times Y$ to a feature vector $\psi(x, y) \in R^d$. The task is to estimate a parameter vector $\alpha \in R^d$ for the weights of the features. The feature vectors $\psi(x, y)$ can include arbitrary features of parse trees, as we discussed in section 2.1. The loss function used in structure-perceptron learning is usually of form:

$$Loss(\alpha) = \psi(x, y) \cdot \alpha - \max_{\overline{y}} \psi(x, \overline{y}) \cdot \alpha, \tag{8}$$

---

**Input:** A set of training images with ground truth $(x^i, y^i)$ for $i = 1..N$. Initialize parameter vector $\alpha = 0$.
For $t = 1..T$, $i = 1..N$

- find the best state of the model on the i'th training image with current parameter setting, i.e., $y^* = \arg\max_y \psi(x^i, y) \cdot \alpha$

- Update the parameters: $\alpha = \alpha + \psi(x^i, y^i) - \psi(x^i, y^*)$

- Store: $\alpha^{t,i} = \alpha$

**Output:** Parameters $\gamma = \sum_{t,i} \alpha^{t,i}/NT$

---

Figure 3: Structure-perceptron learning

where $y$ is the correct structure for input $x$, and $\overline{y}$ is a dummy variable.

The basic structure-perceptron algorithm is designed to minimize the loss function. We adapt *"the averaged parameters"* version whose pseudo-code is given in figure 3. The algorithm proceeds in a simple way (similar to the perceptron algorithm for classification). The parameters are initialized to zero and the algorithm loops over the training examples. If the highest scoring parse tree for input $x$ is not correct, then the parameters $\alpha$ are updated by an additive term. The most difficult step of the method is finding $y^* = \arg\max_y \psi(x^i, y) \cdot \alpha$. This is precisely the parsing (inference) problem. Hence the practicality of structure-perceptron learning, and its computational efficiency, depends on the inference algorithm. As discussed earlier, see section 2.2, the inference algorithm has polynomial computational complexity for an HIM which makes structure-perceptron learning practical for HIM. The averaged parameters are defined to be $\gamma = \sum_{t=1}^{T} \sum_{i=1}^{N} \alpha^{t,i}/NT$, where $T$ is the number of epochs, $NT$ is the total number of iterations. It is straightforward to store these averaged parameters and output them as the final estimates.

## 3 Experimental Results

**Dataset.** We use a standard public dataset, the MSRC 21-class Image Dataset [8], to perform experimental evaluations for the HIM. This dataset is designed to evaluate scene labeling including both image segmentation and multi-class object recognition. The ground truth only gives the labeling of the image pixels. To supplement this ground truth (to enable learning), we estimate the true labels (states of the S-R pair ) of the nodes in the five-layer hierarchy of HIM by selecting the S-R pairs which have maximum overlap with the labels of the image pixels. This approximation only results in $2\%$ error in labeling image pixels. There are a total of $591$ images. We use the identical splitting as [8], i.e., $45\%$ for training, $10\%$ for validation, and $45\%$ for testing. The parameters learnt from the training set, with the best performance on validation set, are selected.

**Implementation Details**. For a given image $x$, the parsing result is obtained by estimating the best configuration $y^*$ of the HIM. To evaluate the performance of parsing we use the **global accuracy** measured in terms of all pixels and the **average accuracy** over the 21 object classes (global accuracy pays most attention to frequently occurring objects and penalizes infrequent objects). A computer with 8 GB memory and 2.4 GHz CPU was used for training and testing. For each class, there are around $4, 500$ weak classifiers selected by multi-class boosting. The boosting learning takes about 35 hours of which 27 hours are spent on I/O processing and 8 hours on computing. The structure-perceptron learning takes about 20 hours to converge in $5520(T = 20, N = 276)$ iterations. In the testing stage, it takes 30 seconds to parse an image with size of $320 \times 200$ (6s for extracting image features, 9s for computing the strong classifier of boosting and 15s for parsing the HIM).

**Results.** Figure 4 (best viewed in color) shows several parsing results obtained by the HIM and by the classifier by itself (i.e. $p(o_a^r|x)$ learnt by boosting). One can see that the HIM is able to roughly capture different shaped segmentation boundaries (see the legs of the cow and sheep in rows 1 and 3, and the boundary curve between sky and building in row 4). Table 1 shows that HIM improves the results obtained by the classifier by $6.9\%$ for average accuracy and $5.3\%$ for global accuracy. In particular, in rows 6 and 7 in figure 4, one can observe that boosting gives many incorrect labels. It is impossible to correct such large mislabeled regions without the long-range interactions in the HIM, which improves the results by $20\%$ and $32\%$.

**Comparisons.** In table 1, we compare the performance of our approach with other successful methods [8, 20, 21]. Our approach outperforms those alternatives by $6\%$ in average accuracy and $4\%$ in global accuracy. Our boosting results are better than Textonboost [8] because of image features. Would we get better results if we use a flat CRF with our boosting instead of a hierarchy? We argue that we would not because the CRF only improves TextonBoost's performance by 3 percent [8], while we gain 5 percent by using the hierarchy (and we start with a higher baseline). Some other

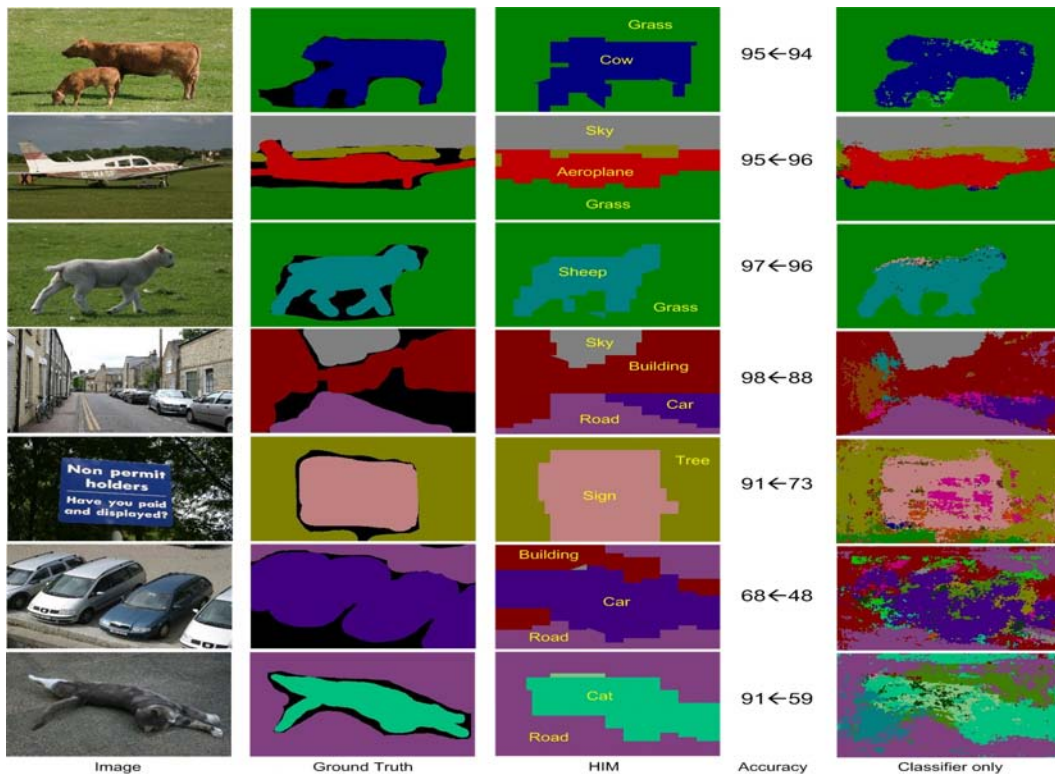

Figure 4: This figure is best viewed in color. The colors indicate the labels of 21 object classes as in [8]. The columns (except the fourth "accuracy" column) show the input images, ground truth, the labels obtained by HIM and the boosting classifier respectively. The "accuracy" column shows the global accuracy obtained by HIM (left) and the boosting classifier (right). In these 7 examples, HIM improves boosting by 1%, -1% (an outlier!), 1%, 10%, 18%, 20% and 32% in terms of global accuracy.

|  | Textonboost[8] | PLSA-MRF [20] | Auto-context [21] | Classifier only | HIM |
|---|---|---|---|---|---|
| Average | 57.7 | 64.0 | 68 | 67.2 | 74.1 |
| Global | 72.2 | 73.5 | 77.7 | 75.9 | 81.2 |

Table 1: Performance Comparisons for average accuracy and global accuracy. "Classifier only" are the results where the pixel labels are predicted by the classifier obtained by boosting only.

methods [22, 11, 10], which are worse than [20, 21] and evaluated on simpler datasets [10, 11] (less than 10 classes), are not listed here due to lack of space. In summary, our results are significantly better than the state-of-the-art methods.

**Diagnosis on the function of S-R Pair.** Figure 5 shows how the S-R pairs (which include the segmentation templates) can be used to (partially) parse an object into its constituent parts, by the correspondence between S-R pairs and specific parts of objects. We plot the states of a subset of S-R pairs for some images. For example, the S-R pair consisting of two horizontal bars labeled "cow" and "grass" respectively indicates the cow's stomach consistently across different images. Similarly, the cow's tail can be located according to the configuration of another S-R pair with vertical bars. In principle, the whole object can be parsed into its constituent parts which are aligned consistently. Developing this idea further is an exciting aspect of our current research.

## 4 Conclusion

This paper describes a novel hierarchical image model (HIM) for 2D image parsing. The hierarchical nature of the model, and the use of recursive segmentation and recognition templates, enables the HIM to represent complex image structures in a coarse-to-fine manner. We can perform inference (parsing) rapidly in polynomial time by exploiting the hierarchical structure. Moreover, we can learn the HIM probability distribution from labeled training data by adapting the structure-perceptron algorithm. We demonstrated the effectiveness of HIM's by applying them to the challenging task of segmentation and labeling of the public MSRC image database. Our results show that we outperform other state-of-the-art approaches.

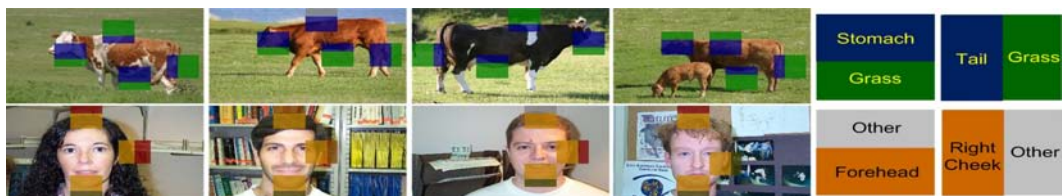

**Figure 5:** The S-R pairs can be used to parse the object into parts. The colors indicate the identities of objects. The shapes (spacial layout) of the segmentation templates distinguish the constituent parts of the object. Observe that the same S-R pairs (e.g. stomach above grass, and tail to the left of grass) correspond to the same object part in different images.

The design of the HIM was motivated by drawing parallels between language and vision processing. We have attempted to capture the underlying spirit of the successful language processing approaches – the hierarchical representations based on the recursive composition of constituents and efficient inference and learning algorithms. Our current work attempts to extend the HIM's to improve their representational power while maintaining computational efficiency.

## 5 Acknowledgments

This research was supported by NSF grant 0413214 and the W.M. Keck foundation.

## References

[1] F. Jelinek and J. D. Lafferty, "Computation of the probability of initial substring generation by stochastic context-free grammars," *Computational Linguistics*, vol. 17, no. 3, pp. 315–323, 1991.

[2] M. Collins, "Head-driven statistical models for natural language parsing," *Ph.D. Thesis, University of Pennsylvania*, 1999.

[3] K. Lari and S. J. Young, "The estimation of stochastic context-free grammars using the inside-outside algorithm," in *Computer Speech and Languag*, 1990.

[4] M. Shilman, P. Liang, and P. A. Viola, "Learning non-generative grammatical models for document analysis," in *Proceedings of IEEE International Conference on Computer Vision*, 2005, pp. 962–969.

[5] Z. Tu and S. C. Zhu, "Image segmentation by data-driven markov chain monte carlo," *IEEE Transactions on Pattern Analysis and Machine Intelligence*, vol. 24, no. 5, pp. 657–673, 2002.

[6] Z. Tu, X. Chen, A. L. Yuille, and S. C. Zhu, "Image parsing: Unifying segmentation, detection, and recognition," in *Proceedings of IEEE International Conference on Computer Vision*, 2003, pp. 18–25.

[7] J. D. Lafferty, A. McCallum, and F. C. N. Pereira, "Conditional random fields: Probabilistic models for segmenting and labeling sequence data," in *Proceedings of International Conference on Machine Learning*, 2001, pp. 282–289.

[8] J. Shotton, J. M. Winn, C. Rother, and A. Criminisi, "TextonBoost: Joint appearance, shape and context modeling for multi-class object recognition and segmentation," in *Proceedings of European Conference on Computer Vision*, 2006, pp. 1–15.

[9] M. Collins, "Discriminative training methods for hidden markov models: theory and experiments with perceptron algorithms," in *Proceedings of Annual Meeting on Association for Computational Linguistics conference on Empirical methods in natural language processing*, 2002, pp. 1–8.

[10] X. He, R. S. Zemel, and M. Á. Carreira-Perpiñán, "Multiscale conditional random fields for image labeling," in *Proceedings of IEEE Computer Society Conference on Computer Vision and Pattern Recognition*, 2004, pp. 695–702.

[11] S. Kumar and M. Hebert, "A hierarchical field framework for unified context-based classification," in *Proceedings of IEEE International Conference on Computer Vision*, 2005, pp. 1284–1291.

[12] E. L. Allwein, R. E. Schapire, and Y. Singer, "Reducing multiclass to binary: A unifying approach for margin classifiers," *Journal of Machine Learning Research*, vol. 1, pp. 113–141, 2000.

[13] Y. Boykov and M.-P. Jolly, "Interactive graph cuts for optimal boundary and region segmentation of objects in n-d images," in *Proceedings of IEEE International Conference on Computer Vision*, 2001, pp. 105–112.

[14] A. Oliva and A. Torralba, "Building the gist of a scene: the role of global image features in recognition," *IEEE Transactions on Pattern Analysis and Machine Intelligence*, vol. 155, pp. 23–36, 2006.

[15] A. Levin and Y. Weiss, "Learning to combine bottom-up and top-down segmentation," in *Proceedings of European Conference on Computer Vision*, 2006, pp. 581–594.

[16] E. B. Sudderth, A. B. Torralba, W. T. Freeman, and A. S. Willsky, "Learning hierarchical models of scenes, objects, and parts," in *Proceedings of IEEE International Conference on Computer Vision*, 2005, pp. 1331–1338.

[17] Y. Chen, L. Zhu, C. Lin, A. L. Yuille, and H. Zhang, "Rapid inference on a novel and/or graph for object detection, segmentation and parsing," in *Advances in Neural Information Processing Systems*, 2007.

[18] B. Taskar, D. Klein, M. Collins, D. Koller, and C. Manning, "Max-margin parsing," in *Proceedings of Annual Meeting on Association for Computational Linguistics conference on Empirical methods in natural language processing*, 2004.

[19] L. Zhu, Y. Chen, X. Ye, and A. L. Yuille, "Structure-perceptron learning of a hierarchical log-linear model," in *Proceedings of IEEE Computer Society Conference on Computer Vision and Pattern Recognition*, 2008.

[20] J. Verbeek and B. Triggs, "Region classification with markov field aspect models," in *Proceedings of IEEE Computer Society Conference on Computer Vision and Pattern Recognition*, 2007.

[21] Z. Tu, "Auto-context and its application to high-level vision tasks," in *Proceedings of IEEE Computer Society Conference on Computer Vision and Pattern Recognition*, 2008.

[22] J. Verbeek and B. Triggs, "Scene segmentation with crfs learned from partially labeled images," in *Advances in Neural Information Processing Systems*, vol. 20, 2008.

